# Conditional Random Sampling: A Sketch-based Sampling Technique for Sparse Data

**Ping Li**
Department of Statistics
Stanford University
Stanford, CA 94305
pingli@stat.stanford.edu

**Kenneth W. Church**
Microsoft Research
One Microsoft Way
Redmond, WA 98052
church@microsoft.com

**Trevor J. Hastie**
Department. of Statistics
Stanford University
Stanford, CA 94305
hastie@stanford.edu

## Abstract

We[1] develop *Conditional Random Sampling (CRS)*, a technique particularly suitable for sparse data. In large-scale applications, the data are often highly sparse. CRS combines sketching and sampling in that it converts sketches of the data into *conditional random samples* online in the estimation stage, with the sample size determined retrospectively. This paper focuses on approximating pairwise $l_2$ and $l_1$ distances and comparing CRS with random projections. For boolean (0/1) data, CRS is provably better than random projections. We show using real-world data that CRS often outperforms random projections. This technique can be applied in learning, data mining, information retrieval, and database query optimizations.

## 1 Introduction

*Conditional Random Sampling (CRS)* is a sketch-based sampling technique that effectively exploits data sparsity. In modern applications in learning, data mining, and information retrieval, the datasets are often very large and also highly sparse. For example, the *term-document* matrix is often more than $99\%$ sparse [7]. Sampling large-scale sparse data is challenging. The *conventional random sampling* (i.e., randomly picking a small fraction) often performs poorly when most of the samples are zeros. Also, in heavy-tailed data, the estimation errors of random sampling could be very large.

As alternatives to random sampling, various *sketching* algorithms have become popular, e.g., random projections [17] and min-wise sketches [6]. Sketching algorithms are designed for approximating *specific* summary statistics. For a specific task, a sketching algorithm often outperforms random sampling. On the other hand, random sampling is much more flexible. For example, we can use the same set of random samples to estimate any $l_p$ pairwise distances and multi-way associations. *Conditional Random Sampling (CRS)* combines the advantages of both sketching and random sampling.

Many important applications concern only the pairwise distances, e.g., distance-based clustering and classification, multi-dimensional scaling, kernels. For a large training set (e.g., at Web scale), computing pairwise distances exactly is often too time-consuming or even infeasible.

Let $\mathbf{A}$ be a data matrix of $n$ rows and $D$ columns. For example, $\mathbf{A}$ can be the *term-document* matrix with $n$ as the total number of word types and $D$ as the total number of documents. In modern search engines, $n \approx 10^6 \sim 10^7$ and $D \approx 10^{10} \sim 10^{11}$. In general, $n$ is the number of data points and $D$ is the number of features. Computing all pairwise associations $\mathbf{A}\mathbf{A}^{\mathrm{T}}$, also called the *Gram matrix* in machine learning, costs $O(n^2 D)$, which could be daunting for large $n$ and $D$. Various sampling methods have been proposed for approximating Gram matrix and kernels [2, 8]. For example, using (normal) random projections [17], we approximate $\mathbf{A}\mathbf{A}^{\mathrm{T}}$ by $(\mathbf{A}\mathbf{R})(\mathbf{A}\mathbf{R})^{\mathrm{T}}$, where the entries of $\mathbf{R} \in \mathbb{R}^{D \times k}$ are i.i.d. $N(0,1)$. This reduces the cost down to $O(nDk+n^2 k)$, where $k \ll \min(n, D)$.

Sampling techniques can be critical in databases and information retrieval. For example, the database query optimizer seeks highly efficient techniques to estimate the intermediate join sizes in order to choose an "optimum" execution path for multi-way joins.

Conditional Random Sampling (CRS) can be applied to estimating pairwise distances (in any norm) as well as multi-way associations. CRS can also be used for estimating joint histograms (two-way and multi-way). While this paper focuses on estimating pairwise $l_2$ and $l_1$ distances and inner products, we refer readers to the technical report [13] for estimating joint histograms. Our early work, [11, 12] concerned estimating two-way and multi-way associations in boolean (0/1) data.

We will compare CRS with *normal random projections* for approximating $l_2$ distances and inner products, and with *Cauchy random projections* for approximating $l_1$ distances. In boolean data, CRS bears some similarity to *Broder's sketches* [6] with some important distinctions. [12] showed that in boolean data, CRS improves Broder's sketches by roughly halving the estimation variances.

## 2 The Procedures of CRS

*Conditional Random Sampling* is a two-stage procedure. In the *sketching* stage, we scan the data matrix once and store a fraction of the non-zero elements in each data point, as "sketches." In the *estimation* stage, we generate *conditional random samples* online pairwise (for two-way) or group-wise (for multi-way); hence we name our algorithm *Conditional Random Sampling (CRS)*.

### 2.1 The Sampling/Sketching Procedure

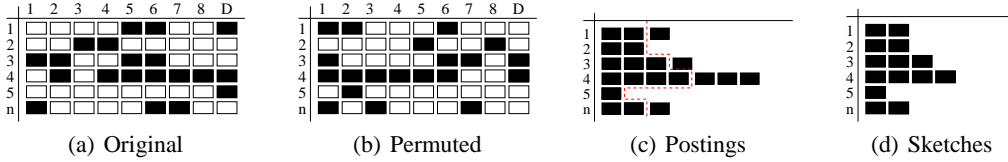

(a) Original      (b) Permuted      (c) Postings      (d) Sketches

Figure 1: A global view of the *sketching* stage.

Figure 1 provides a global view of the *sketching* stage. The columns of a sparse data matrix (a) are first randomly permuted (b). Then only the non-zero entries are considered, called *postings* (c). *Sketches* are simply the front of postings (d). Note that in the actual implementation, we only need to maintain a permutation mapping on the column IDs.

|      | 1 | 2 | 3 | 4 | 5 | 6 | 7 | 8 | 9 | 10 | 11 | 12 | 13 | 14 | 15 |
|------|---|---|---|---|---|---|---|---|---|----|----|----|----|----|----|
| $u_1$ | 0 | 1 | 0 | 2 | 0 | 1 | 0 | 0 | 1 | 2  | 1  | 0  | 1  | 0  | 2  |
| $u_2$ | 1 | 3 | 0 | 0 | 1 | 2 | 0 | 1 | 0 | 0  | 3  | 0  | 0  | 2  | 1  |

(a) Data matrix and random samples

$P_1$ :  2 (1)  4 (2)  6 (1)  9 (1)  10 (2)  11 (1)  13 (1)  15 (2)

$P_2$:  1 (1)  2 (3)  5 (1)  6 (2)  8 (1)  11 (3)  14 (2)  15 (1)

$K_1$ :  2 (1)  4 (2)  6 (1)  9 (1)  10 (2)

$K_2$:  1 (1)  2 (3)  5 (1)  6 (2)  8 (1)  11 (3)

(b) Postings                    (c) Sketches

Figure 2: (a): A data matrix with two rows and $D = 15$. If the column IDs are random, the first $D_s = 10$ columns constitute a random sample. $u_i$ denotes the $i$th row. (b): Postings consist of tuples "ID (Value)." (c): Sketches are the first $k_i$ entries of postings sorted ascending by IDs. In this example, $k_1 = 5$, $k_2 = 6$, $D_s = \min(10, 11) = 10$. Excluding 11(3) in $K_2$, we obtain the same samples as if we directly sampled the first $D_s = 10$ columns in the data matrix.

Apparently sketches are not uniformly random samples, which may make the estimation task difficult. We show, in Figure 2, that sketches are almost random samples pairwise (or group-wise). Figure 2(a) constructs *conventional random samples* from a data matrix; and we show one can generate (retrospectively) the same random samples from sketches in Figure 2(b)(c).

In Figure 2(a), when the column are randomly permuted, we can construct random samples by simply taking the first $D_s$ columns from the data matrix of $D$ columns ($D_s \ll D$ in real applications).

For sparse data, we only store the non-zero elements in the form of tuples "ID (Value)," a structure called *postings*. We denote the postings by $P_i$ for each row $u_i$. Figure 2(b) shows the postings for the same data matrix in Figure 2(a). The tuples are sorted ascending by their IDs. A *sketch*, $K_i$, of postings $P_i$, is the first $k_i$ entries (i.e., the smallest $k_i$ IDs) of $P_i$, as shown in Figure 2(c).

The central observation is that if we exclude all elements of sketches whose IDs are larger than

$$D_s = \min \left( \max(\text{ID}(K_1)), \max(\text{ID}(K_2)) \right), \quad (1)$$

we obtain exactly the same samples as if we directly sampled the first $D_s$ columns from the data matrix in Figure 2(a). This way, we convert sketches into random samples by *conditioning* on $D_s$, which differs pairwise and we do not know beforehand.

## 2.2 The Estimation Procedure

The estimation task for CRS can be extremely simple. After we construct the conditional random samples from sketches $K_1$ and $K_2$ with the effective sample size $D_s$, we can compute any distances ($l_2$, $l_1$, or inner products) from the samples and multiply them by $\frac{D}{D_s}$ to estimate the original space. (Later, we will show how to improve the estimates by taking advantage of the marginal information.)

We use $\tilde{u}_{1,j}$ and $\tilde{u}_{2,j}$ ($j = 1$ to $D_s$) to denote the conditional random samples (of size $D_s$) obtained by CRS. For example, in Figure 2, we have $D_s = 10$, and the non-zero $\tilde{u}_{1,j}$ and $\tilde{u}_{2,j}$ are

$$\tilde{u}_{1,2} = 3, \ \tilde{u}_{1,4} = 2, \ \tilde{u}_{1,6} = 1, \ \tilde{u}_{1,9} = 1, \ \tilde{u}_{1,10} = 2$$
$$\tilde{u}_{2,1} = 1, \ \tilde{u}_{2,2} = 3, \ \tilde{u}_{2,5} = 1, \ \tilde{u}_{2,6} = 2, \ \tilde{u}_{2,8} = 1.$$

Denote the inner product, squared $l_2$ distance, and $l_1$ distance, by $a$, $d^{(2)}$, and $d^{(1)}$, respectively,

$$a = \sum_{i=1}^{D} u_{1,i} u_{2,i}, \quad d^{(2)} = \sum_{i=1}^{D} |u_{1,i} - u_{2,i}|^2, \quad d^{(1)} = \sum_{i=1}^{D} |u_{1,i} - u_{2,i}| \quad (2)$$

Once we have the random samples, we can then use the following simple linear estimators:

$$\hat{a}_{MF} = \frac{D}{D_s} \sum_{j=1}^{D_s} \tilde{u}_{1,j} \tilde{u}_{2,j}, \quad \hat{d}_{MF}^{(2)} = \frac{D}{D_s} \sum_{j=1}^{D_s} \left( \tilde{u}_{1,j} - \tilde{u}_{2,j} \right)^2, \quad \hat{d}_{MF}^{(1)} = \frac{D}{D_s} \sum_{j=1}^{D_s} |\tilde{u}_{1,j} - \tilde{u}_{2,j}|. \quad (3)$$

## 2.3 The Computational Cost

Th sketching stage requires generating a random permutation mapping of length $D$, and linear scan all the non-zeros. Therefore, generating sketches for $\mathbf{A} \in \mathbb{R}^{n \times D}$ costs $O(\sum_{i=1}^{n} f_i)$, where $f_i$ is the number of non-zeros in the $i$th row, i.e., $f_i = |P_i|$. In the estimation stage, we need to linear scan the sketches. While the conditional sample size $D_s$ might be large, the cost for estimating the distance between one pair of data points would be only $O(k_1 + k_2)$ instead of $O(D_s)$.

# 3 The Theoretical Variance Analysis of CRS

We give some theoretical analysis on the variances of CRS. For simplicity, we ignore the "finite population correction factor", $\frac{D - D_s}{D - 1}$, due to "sample-without-replacement."

We first consider $\hat{a}_{MF} = \frac{D}{D_s} \sum_{j=1}^{D_s} \tilde{u}_{1,j} \tilde{u}_{2,j}$. By assuming "sample-with-replacement," the samples, $(\tilde{u}_{1,j} \tilde{u}_{2,j})$, $j = 1$ to $D_s$, are i.i.d, conditional on $D_s$. Thus,

$$\text{Var}(\hat{a}_{MF} | D_s) = \left( \frac{D}{D_s} \right)^2 D_s \text{Var} \left( \tilde{u}_{1,1} \tilde{u}_{2,1} \right) = \frac{D}{D_s} D \left( \text{E} \left( \tilde{u}_{1,1} \tilde{u}_{2,1} \right)^2 - \text{E}^2 \left( \tilde{u}_{1,1} \tilde{u}_{2,1} \right) \right), \quad (4)$$

$$\text{E} \left( \tilde{u}_{1,1} \tilde{u}_{2,1} \right) = \frac{1}{D} \sum_{i=1}^{D} \left( u_{1,i} u_{2,i} \right) = \frac{a}{D}, \qquad \text{E} \left( \tilde{u}_{1,1} \tilde{u}_{2,1} \right)^2 = \frac{1}{D} \sum_{i=1}^{D} \left( u_{1,i} u_{2,i} \right)^2, \quad (5)$$

$$\text{Var}(\hat{a}_{MF} | D_s) = \frac{D}{D_s} D \left( \frac{1}{D} \sum_{i=1}^{D} \left( u_{1,i} u_{2,i} \right)^2 - \left( \frac{a}{D} \right)^2 \right) = \frac{D}{D_s} \left( \sum_{i=1}^{D} u_{1,i}^2 u_{2,i}^2 - \frac{a^2}{D} \right). \quad (6)$$

The unconditional variance would be simply

$$\text{Var}(\hat{a}_{MF}) = \text{E} \left( \text{Var}(\hat{a}_{MF} | D_s) \right) = \text{E} \left( \frac{D}{D_s} \right) \left( \sum_{i=1}^{D} u_{1,i}^2 u_{2,i}^2 - \frac{a^2}{D} \right),$$

as $\text{Var}(\hat{X}) = \text{E}\left(\text{Var}(\hat{X}|D_s)\right) + \text{Var}\left(\text{E}(\hat{X}|D_s)\right) = \text{E}\left(\text{Var}(\hat{X}|D_s)\right)$, when $\hat{X}$ is conditionally unbiased.

No closed-form expression is known for $\text{E}\left(\frac{D}{D_s}\right)$; but we know $\text{E}\left(\frac{D}{D_s}\right) \geq \max\left(\frac{f_1}{k_1}, \frac{f_2}{k_2}\right)$ (similar to Jensen's inequality). Asymptotically (as $k_1$ and $k_2$ increase), the inequality becomes an equality

$$\text{E}\left(\frac{D}{D_s}\right) \approx \max\left(\frac{f_1+1}{k_1}, \frac{f_2+1}{k_2}\right) \approx \max\left(\frac{f_1}{k_1}, \frac{f_2}{k_2}\right), \tag{7}$$

where $f_1$ and $f_2$ are the numbers of non-zeros in $u_1$ and $u_2$, respectively. See [13] for the proof. Extensive simulations in [13] verify that the errors of (7) are usually within $5\%$ when $k_1, k_2 > 20$.

We similarly derive the variances for $\hat{d}_{MF}^{(2)}$ and $\hat{d}_{MF}^{(1)}$. In a summary, we obtain (when $k_1 = k_2 = k$)

$$\text{Var}\left(\hat{a}_{MF}\right) = \text{E}\left(\frac{D}{D_s}\right)\left(\sum_{i=1}^{D} u_{1,i}^2 u_{2,i}^2 - \frac{a^2}{D}\right) \approx \frac{\max(f_1, f_2)}{D} \frac{1}{k}\left(D\sum_{i=1}^{D} u_{1,i}^2 u_{2,i}^2 - a^2\right), \tag{8}$$

$$\text{Var}\left(\hat{d}_{MF}^{(2)}\right) = \text{E}\left(\frac{D}{D_s}\right)\left(d^{(4)} - \frac{[d^{(2)}]^2}{D}\right) \approx \frac{\max(f_1, f_2)}{D} \frac{1}{k}\left(Dd^{(4)} - [d^{(2)}]^2\right), \tag{9}$$

$$\text{Var}\left(\hat{d}_{MF}^{(1)}\right) = \text{E}\left(\frac{D}{D_s}\right)\left(d^{(2)} - \frac{[d^{(1)}]^2}{D}\right) \approx \frac{\max(f_1, f_2)}{D} \frac{1}{k}\left(Dd^{(2)} - [d^{(1)}]^2\right). \tag{10}$$

where we denote $d^{(4)} = \sum_{i=1}^{D}\left(u_{1,i} - u_{2,i}\right)^4$.

The *sparsity* term $\frac{\max(f_1, f_2)}{D}$ reduces the variances significantly. If $\frac{\max(f_1, f_2)}{D} = 0.01$, the variances can be reduced by a factor of 100, compared to *conventional random coordinate sampling*.

## 4 A Brief Introduction to Random Projections

We give a brief introduction to random projections, with which we compare CRS. (Normal) Random projections [17] are widely used in learning and data mining [2–4].

Random projections multiply the data matrix $\mathbf{A} \in \mathbb{R}^{n \times D}$ with a random matrix $\mathbf{R} \in \mathbb{R}^{D \times k}$ to generate a compact representation $\mathbf{B} = \mathbf{AR} \in \mathbb{R}^{n \times k}$. For estimating $l_2$ distances, $\mathbf{R}$ typically consists of i.i.d. entries in $N(0, 1)$; hence we call it *normal random projections*. For $l_1$, $\mathbf{R}$ consists of i.i.d. Cauchy $C(0, 1)$ [9]. However, the recent impossibility result [5] has ruled out estimators that could be metrics for dimension reduction in $l_1$.

Denote $v_1, v_2 \in \mathbb{R}^k$ the two rows in $\mathbf{B}$, corresponding to the original data points $u_1, u_2 \in \mathbb{R}^D$. We also introduce the notation for the marginal $l_2$ norms: $m_1 = \|u_1\|^2$, $m_2 = \|u_2\|^2$.

### 4.1 Normal Random Projections

In this case, $\mathbf{R}$ consists of i.i.d. $N(0, 1)$. It is easy to show that the following linear estimators of the inner product $a$ and the squared $l_2$ distance $d^{(2)}$ are unbiased

$$\hat{a}_{NRP,MF} = \frac{1}{k} v_1^\text{T} v_2, \qquad \hat{d}_{NRP,MF}^{(2)} = \frac{1}{k}\|v_1 - v_2\|^2, \tag{11}$$

with variances [15, 17]

$$\text{Var}\left(\hat{a}_{NRP,MF}\right) = \frac{1}{k}\left(m_1 m_2 + a^2\right), \qquad \text{Var}\left(\hat{d}_{NRP,MF}^{(2)}\right) = \frac{2[d^{(2)}]^2}{k}. \tag{12}$$

Assuming that the margins $m_1 = \|u_1\|^2$ and $m_2 = \|u_2\|^2$ are known, [15] provides a maximum likelihood estimator, denoted by $\hat{a}_{NRP,MLE}$, whose (asymptotic) variance is

$$\text{Var}\left(\hat{a}_{NRP,MLE}\right) = \frac{1}{k}\frac{(m_1 m_2 - a^2)^2}{m_1 m_2 + a^2} + O(k^{-2}). \tag{13}$$

### 4.2 Cauchy Random Projections for Dimension Reduction in $l_1$

In this case, $\mathbf{R}$ consisting of i.i.d. entries in Cauchy $C(0, 1)$. [9] proposed an estimator based on the absolute sample median. Recently, [14] proposed a variety of nonlinear estimators, including, a bias-corrected sample median estimator, a bias-corrected geometric mean estimator, and a bias-corrected

maximum likelihood estimator. An analog of the Johnson-Lindenstrauss (JL) lemma for dimension reduction in $l_1$ is also proved in [14], based on the bias-corrected geometric mean estimator.

We only list the maximum likelihood estimator derived in [14], because it is the most accurate one.

$$\hat{d}^{(1)}_{CRP,MLE,c} = \left(1 - \frac{1}{k}\right)\hat{d}^{(1)}_{CRP,MLE}, \tag{14}$$

where $\hat{d}^{(1)}_{CRP,MLE}$ solves a nonlinear MLE equation

$$-\frac{k}{\hat{d}^{(1)}_{CRP,MLE}} + \sum_{j=1}^{k} \frac{2\hat{d}^{(1)}_{CRP,MLE}}{(v_{1,j} - v_{2,j})^2 + \left(\hat{d}^{(1)}_{CRP,MLE}\right)^2} = 0. \tag{15}$$

[14] shows that

$$\text{Var}\left(\hat{d}^{(1)}_{CRP,MLE,c}\right) = \frac{2[d^{(1)}]^2}{k} + \frac{3[d^{(1)}]^2}{k^2} + O\left(\frac{1}{k^3}\right). \tag{16}$$

### 4.3 General Stable Random Projections for Dimension Reduction in $l_p$ ($0 < p \leq 2$)

[10] generalized the bias-corrected geometric mean estimator to general stable random projections for dimension reduction in $l_p$ ($0 < p \leq 2$), and provided the theoretical variances and exponential tail bounds. Of course, CRS can also be applied to approximating any $l_p$ distances.

## 5 Improving CRS Using Marginal Information

It is often reasonable to assume that we know the marginal information such as marginal $l_2$ norms, numbers of non-zeros, or even marginal histograms. This often leads to (much) sharper estimates, by maximizing the likelihood under marginal constraints. In the boolean data case, we can express the MLE solution explicitly and derive a closed-form (asymptotic) variance. In general real-valued data, the joint likelihood is not available; we propose an approximate MLE solution.

### 5.1 Boolean (0/1) Data

In 0/1 data, estimating the inner product becomes estimating a two-way contingency table, which has four cells. Because of the margin constraints, there is only one degree of freedom. Therefore, it is not hard to show that the MLE of $a$ is the solution, denoted by $\hat{a}_{0/1,MLE}$, to a cubic equation

$$\frac{s_{11}}{a} - \frac{s_{10}}{f_1 - a} - \frac{s_{01}}{f_2 - a} + \frac{s_{00}}{D - f_1 - f_2 + a} = 0, \tag{17}$$

where $s_{11} = \#\{j : \tilde{u}_{1,j} = \tilde{u}_{2,j} = 1\}$, $s_{10} = \#\{j : \tilde{u}_{1,j} = 1, \tilde{u}_{2,j} = 0\}$, $s_{01} = \#\{j : \tilde{u}_{1,j} = 0, \tilde{u}_{2,j} = 1\}$, $s_{00} = \#\{j : \tilde{u}_{1,j} = 0, \tilde{u}_{2,j} = 0\}$, $j = 1, 2, ..., D_s$.

The (asymptotic) variance of $\hat{a}_{0/1,MLE}$ is proved [11–13] to be

$$\text{Var}(\hat{a}_{0/1,MLE}) = \text{E}\left(\frac{D}{D_s}\right)\frac{1}{\frac{1}{a} + \frac{1}{f_1-a} + \frac{1}{f_2-a} + \frac{1}{D-f_1-f_2+a}}. \tag{18}$$

### 5.2 Real-valued Data

A practical solution is to assume some parametric form of the (bivariate) data distribution based on prior knowledge; and then solve an MLE considering various constraints. Suppose the samples $(\tilde{u}_{1,j}, \tilde{u}_{2,j})$ are i.i.d. bivariate normal with moments determined by the population moments, i.e.,

$$\begin{bmatrix} \tilde{v}_{1,j} \\ \tilde{v}_{2,j} \end{bmatrix} = \begin{bmatrix} \tilde{u}_{1,j} - \bar{u}_1 \\ \tilde{u}_{2,j} - \bar{u}_2 \end{bmatrix} \sim N\left(\begin{bmatrix} 0 \\ 0 \end{bmatrix}, \tilde{\Sigma}\right), \tag{19}$$

$$\tilde{\Sigma} = \frac{1}{D_s}\frac{D_s}{D}\begin{bmatrix} \|u_1\|^2 - D\bar{u}_1^2 & u_1^T u_2 - D\bar{u}_1\bar{u}_2 \\ u_1^T u_2 - D\bar{u}_1\bar{u}_2 & \|u_2\|^2 - D\bar{u}_2^2 \end{bmatrix} = \frac{1}{D_s}\begin{bmatrix} \ddot{m}_1 & \ddot{a} \\ \ddot{a} & \ddot{m}_2 \end{bmatrix}, \tag{20}$$

where $\bar{u}_1 = \sum_{i=1}^{D} u_{1,i}/D$, $\bar{u}_2 = \sum_{i=1}^{D} u_{2,i}/D$ are the population means. $\ddot{m}_1 = \frac{D_s}{D}\left(\|u_1\|^2 - D\bar{u}_1^2\right)$, $\ddot{m}_2 = \frac{D_s}{D}\left(\|u_2\|^2 - D\bar{u}2^2\right)$, $\ddot{a} = \frac{D_s}{D}\left(u_1^T u_2 - D\bar{u}_1\bar{u}_2\right)$. Suppose that $\bar{u}_1$, $\bar{u}_2$, $m_1 = \|u_1\|^2$ and $m_2 = \|u_2\|^2$ are known, an MLE for $a = u_1^T u_2$, denoted by $\hat{a}_{MLE,N}$, is

$$\hat{a}_{MLE,N} = \frac{D}{D_s}\hat{a} + D\bar{u}_1\bar{u}_2, \tag{21}$$

where, similar to Lemma 2 of [15], $\hat{\hat{a}}$ is the solution to a cubic equation:

$$\dddot{a}^3 - \ddot{a}^2 \left(\tilde{v}_1^T \tilde{v}_2\right) + \ddot{a}\left(-\ddot{m}_1 \ddot{m}_2 + \ddot{m}_1 \|\tilde{v}_2\|^2 + \ddot{m}_2 \|\tilde{v}_1\|^2\right) - \ddot{m}_1 \ddot{m}_2 \tilde{v}_1^T \tilde{v}_2 = 0. \qquad (22)$$

$\hat{a}_{MLE,N}$ is fairly robust, although sometimes we observe the biases are quite noticeable. In general, this is a good bias-variance trade-off (especially when $k$ is not too large). Intuitively, the reason why this (seemly crude) assumption of bivariate normality works well is because, once we have fixed the margins, we have removed to a large extent the non-normal component of the data.

# 6 Theoretical Comparisons of CRS With Random Projections

As reflected by their variances, for general data types, whether CRS is better than random projections depends on two competing factors: data sparsity and data heavy-tailedness. However, in the following two important scenarios, CRS outperforms random projections.

## 6.1 Boolean (0/1) data

In this case, the marginal norms are the same as the numbers of non-zeros, i.e., $m_i = \|u_i\|^2 = f_i$.

Figure 3 plots the ratio, $\frac{\text{Var}(\hat{a}_{MF})}{\text{Var}(\hat{a}_{NRP,MF})}$, verifying that CRS is (considerably) more accurate:

$$\frac{\text{Var}\left(\hat{a}_{MF}\right)}{\text{Var}\left(\hat{a}_{NRP,MF}\right)} = \frac{\max(f_1, f_2)}{f_1 f_2 + a^2} \frac{1}{\frac{1}{a} + \frac{1}{D-a}} \leq \frac{\max(f_1, f_2)a}{f_1 f_2 + a^2} \leq 1.$$

Figure 4 plots $\frac{\text{Var}(\hat{a}_{0/1,MLE})}{\text{Var}(\hat{a}_{NRP,MLE})}$. In most possible range of the data, this ratio is less than 1. When $u_1$ and $u_2$ are very close (e.g., $a \approx f_2 \approx f_1$), random projections appear more accurate. However, when this does occur, the absolute variances are so small (even zero) that their ratio does not matter.

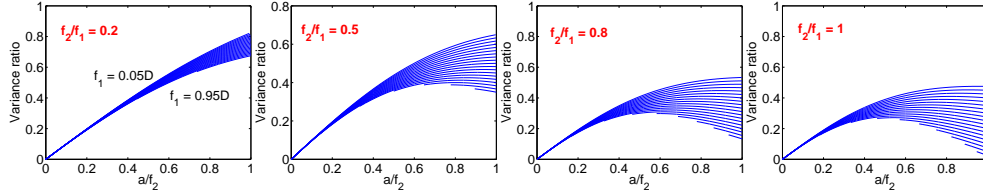

Figure 3: The variance ratios, $\frac{\text{Var}(\hat{a}_{MF})}{\text{Var}(\hat{a}_{NRP,MF})}$, show that CRS has smaller variances than random projections, when no marginal information is used. We let $f_1 \geq f_2$ and $f_2 = \alpha f_1$ with $\alpha = 0.2, 0.5, 0.8, 1.0$. For each $\alpha$, we plot from $f_1 = 0.05D$ to $f_1 = 0.95D$ spaced at $0.05D$.

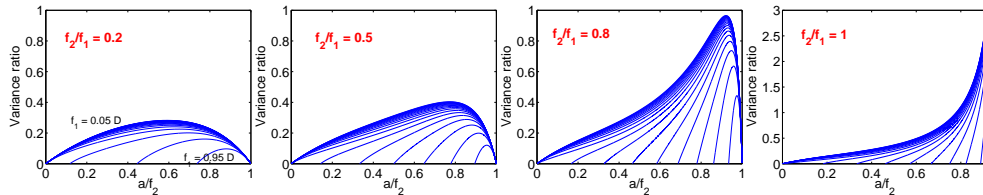

Figure 4: The ratios, $\frac{\text{Var}(\hat{a}_{0/1,MLE})}{\text{Var}(\hat{a}_{NRP,MLE})}$, show that CRS usually has smaller variances than random projections, except when $f_1 \approx f_2 \approx a$.

## 6.2 Nearly Independent Data

Suppose two data points $u_1$ and $u_2$ are independent (or less strictly, uncorrelated to the second order), it is easy to show that the variance of CRS is always smaller:

$$\text{Var}\left(\hat{a}_{MF}\right) \leq \frac{\max(f_1, f_2)}{D} \frac{m_1 m_2}{k} \leq \text{Var}\left(\hat{a}_{NRP,MF}\right) = \frac{m_1 m_2 + a^2}{k}, \qquad (23)$$

even if we ignore the data sparsity. Therefore, CRS will be much better for estimating inner products in nearly independent data. Once we have obtained the inner products, we can infer the $l_2$ distances easily by $d^{(2)} = m_1 + m_2 - 2a$, since the margins, $m_1$ and $m_2$, are easy to obtain exactly.

In high dimensions, it is often the case that most of the data points are only very weakly correlated.

### 6.3 Comparing the Computational Efficiency

As previously mentioned, the cost of constructing sketches for $\mathbf{A} \in \mathbb{R}^{n \times D}$ would be $O(nD)$ (or more precisely, $O(\sum_{i=1}^{n} f_i)$). The cost of (normal) random projections would be $O(nDk)$, which can be reduced to $O(nDk/3)$ using *sparse random projections* [1]. Therefore, it is possible that CRS is considerably more efficient than random projections in the sampling stage.[2]

In the estimation stage, CRS costs $O(2k)$ to compute the sample distance for each pair. This cost is only $O(k)$ in random projections. Since $k$ is very small, the difference should not be a concern.

## 7 Empirical Evaluations

We compare CRS with random projections (RP) using real data, including $n = 100$ randomly sampled documents from the NSF data [7] (sparsity $\approx 1\%$), $n = 100$ documents from the NEWS-GROUP data [4] (sparsity $\approx 1\%$), and one class of the COREL image data ($n = 80$, sparsity $\approx 5\%$). We estimate all pairwise inner products, $l_1$ and $l_2$ distances, using both CRS and RP. For each pair, we obtain 50 runs and average the absolute errors. We compare the median errors and the percentage in which CRS does better than random projections.

The results are presented in Figures 5, 6, 7. In each panel, the dashed curve indicates that we sample each data point with equal sample size ($k$). For CRS, we can adjust the sample size according to the sparsity, reflected by the solid curves. We adjust sample sizes only roughly. The data points are divided into 3 groups according to sparsity. Data in different groups are assigned different sample sizes for CRS. For random projections, we use the average sample size.

For both NSF and NEWSGROUP data, CRS overwhelmingly outperforms RP for estimating inner products and $l_2$ distances (both using the marginal information). CRS also outperforms RP for approximating $l_1$ and $l_2$ distances (without using the margins).

For the COREL data, CRS still outperforms RP for approximating inner products and $l_2$ distances (using the margins). However, RP considerably outperforms CRS for approximating $l_1$ distances and $l_2$ distances (without using the margins). Note that the COREL image data are not too sparse and are considerably more heavy-tailed than the NSF and NEWSGROUP data [13].

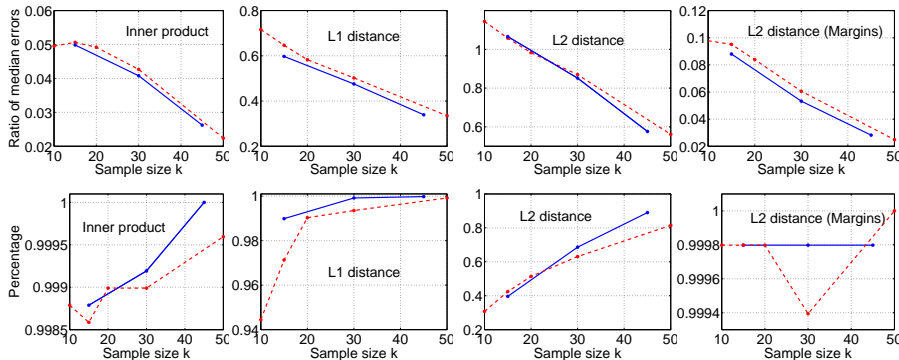

Figure 5: NSF data. Upper four panels: ratios (CRS over RP ( random projections)) of the median absolute errors; values $< 1$ indicate that CRS does better. Bottom four panels: percentage of pairs for which CRS has smaller errors than RP; values $> 0.5$ indicate that CRS does better. Dashed curves correspond to fixed sample sizes while solid curves indicate that we (crudely) adjust sketch sizes in CRS according to data sparsity. In this case, CRS is overwhelmingly better than RP for approximating inner products and $l_2$ distances (both using margins).

## 8 Conclusion

There are many applications of $l_1$ and $l_2$ distances on large sparse datasets. We propose a new sketch-based method, *Conditional Random Sampling (CRS)*, which is provably better than random projections, at least for the important special cases of boolean data and nearly independent data. In general non-boolean data, CRS compares favorably, both theoretically and empirically, especially when we take advantage of the margins (which are easier to compute than distances).

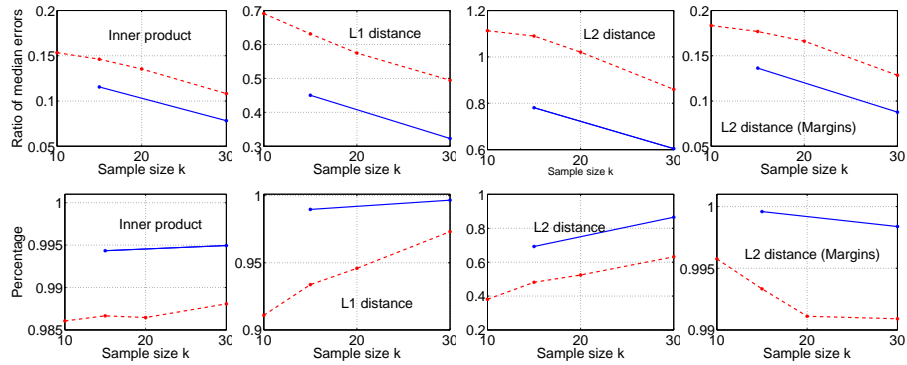

Figure 6: NEWSGROUP data. The results are quite similar to those in Figure 5 for the NSF data. In this case, it is more obvious that adjusting sketch sizes helps CRS.

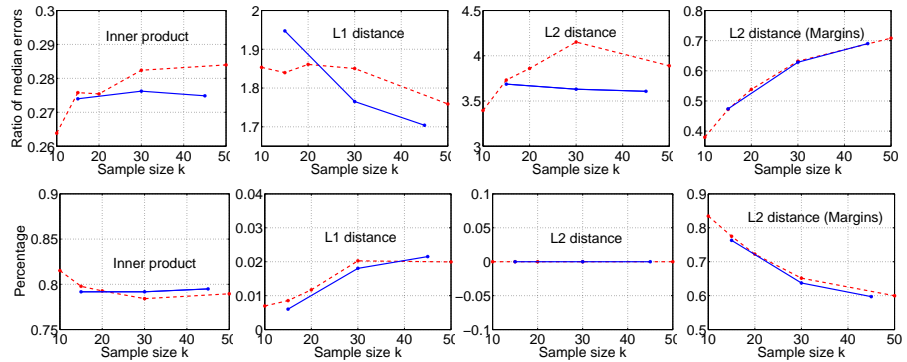

Figure 7: COREL image data.

## Acknowledgment

We thank Chris Burges, David Heckerman, Chris Meek, Andrew Ng, Art Owen, Robert Tibshirani, for various helpful conversations, comments, and discussions. We thank Ella Bingham, Inderjit Dhillon, and Matthias Hein for the datasets.

## Footnotes

[1]The full version [13]: www.stanford.edu/~pingli98/publications/CRS_tr.pdf

[2] [16] proposed *very sparse random projections* to reduce the cost $O(nDk)$ down to $O(n\sqrt{D}k)$.

## References

[1] D. Achlioptas. Database-friendly random projections: Johnson-Lindenstrauss with binary coins. *Journal of Computer and System Sciences*, 66(4):671–687, 2003.

[2] D. Achlioptas, F. McSherry, and B. Schölkopf. Sampling techniques for kernel methods. In *NIPS*, pages 335–342, 2001.

[3] R. Arriaga and S. Vempala. An algorithmic theory of learning: Robust concepts and random projection. *Machine Learning*, 63(2):161–182, 2006.

[4] E. Bingham and H. Mannila. Random projection in dimensionality reduction: Applications to image and text data. In *KDD*, pages 245–250, 2001.

[5] B. Brinkman and M. Charikar. On the impossibility of dimension reduction in $l_1$. *Journal of ACM*, 52(2):766–788, 2005.

[6] A. Broder. On the resemblance and containment of documents. In *the Compression and Complexity of Sequences*, pages 21–29, 1997.

[7] I. Dhillon and D. Modha. Concept decompositions for large sparse text data using clustering. *Machine Learning*, 42(1-2):143–175, 2001.

[8] P. Drineas and M. Mahoney. On the nystrom method for approximating a gram matrix for improved kernel-based learning. *Journal of Machine Learning Research*, 6(Dec):2153–2175, 2005.

[9] P. Indyk. Stable distributions, pseudorandom generators, embeddings and data stream computation. In *FOCS*, pages 189–197, 2000.

[10] P. Li. Very sparse stable random projections, estimators and tail bounds for stable random projections. Technical report, `http://arxiv.org/PS_cache/cs/pdf/0611/0611114.pdf`, 2006.

[11] P. Li and K. Church. Using sketches to estimate associations. In *HLT/EMNLP*, pages 708–715, 2005.

[12] P. Li and K. Church. A sketch algorithm for estimating two-way and multi-way associations. *Computational Linguistics*, To Appear.

[13] P. Li, K. Church, and T. Hastie. Conditional random sampling: A sketched-based sampling technique for sparse data. Technical Report 2006-08, Department of Statistics, Stanford University), 2006.

[14] P. Li, K. Church, and T. Hastie. Nonlinear estimators and tail bounds for dimensional reduction in $l_1$ using Cauchy random projections. (`http://arxiv.org/PS_cache/cs/pdf/0610/0610155.pdf`), 2006.

[15] P. Li, T. Hastie, and K. Church. Improving random projections using marginal information. In *COLT*, pages 635–649, 2006.

[16] P. Li, T. Hastie, and K. Church. Very sparse random projections. In *KDD*, pages 287–296, 2006.

[17] S. Vempala. *The Random Projection Method*. American Mathematical Society, Providence, RI, 2004.
